# Convex Multiple-Instance Learning by Estimating Likelihood Ratio

**Fuxin Li and Cristian Sminchisescu**
Institute for Numerical Simulation, University of Bonn
{fuxin.li,cristian.sminchisescu}@ins.uni-bonn.de

## Abstract

We propose an approach to multiple-instance learning that reformulates the problem as a convex optimization on the likelihood ratio between the positive and the negative class for each training instance. This is casted as joint estimation of both a likelihood ratio predictor and the target (likelihood ratio variable) for instances. Theoretically, we prove a quantitative relationship between the risk estimated under the 0-1 classification loss, and under a loss function for likelihood ratio. It is shown that likelihood ratio estimation is generally a good surrogate for the 0-1 loss, and separates positive and negative instances well. The likelihood ratio estimates provide a ranking of instances within a bag and are used as input features to learn a linear classifier on bags of instances. Instance-level classification is achieved from the bag-level predictions and the individual likelihood ratios. Experiments on synthetic and real datasets demonstrate the competitiveness of the approach.

## 1  Introduction

Multiple Instance Learning (MIL) has been proposed over 10 years ago as a methodology to learn models under weak labeling constraints [1]. Unlike traditional binary classification problems, the positive items are represented as bags, which are sets of instances. A feature vector is used to represent each instance in the bag. There is an OR relationship in a bag: if one of the feature vectors is classified as positive, the entire bag is considered positive. A simple intuition is: one has a number of keys and faces a locked door. To enter the door, we only need one matching keys. MIL is a natural weak labeling formulation for text categorization [2] and computer vision problems [3]. In document classification, one is given files made of many sentences, and often only a few are useful. In computer vision, an image can be decomposed into different regions, and only some delineate objects. Therefore, MIL can be used in sophisticated tasks, such as identifying the location of object parts from bounding box information in images [4]. Although efforts have been made to provide datasets with increasingly more detailed supervisory information [5], without automation such a minutiae level of detail becomes prohibitive for large datasets, or more complicated data like video [6, 7]. In this case, one necessarily needs to resort to multiple-instance learning.

MIL is interesting mainly because of its potential to provide instance-level labels from weak supervisory information. However the state-of-the-art in MIL is often obtained by simply using a weighted sum of kernel values between all instance pairs within the bags, while ignoring the prediction of instance labels [8, 9, 10]. It is intriguing why MIL algorithms that exploit instance level information cannot achieve better performance, as constraints at instance level seems abundant – none of the negative instances is positive. This should provide additional constraints in defining the region of positive instances and should help classification in input space.

A major challenge is the non-convexity of many instance-level MIL algorithms [2, 11, 12, 13, 14]. Most of these algorithms perform alternating minimization on the classifier and the instance weights.

This procedure usually gives only a local optimum since the objective is non-convex. The benchmark performance of MIL methods is overall quite similar, although techniques differ significantly: some assign binary weights to instances [2], some assign real weights [12, 13], yet others use probabilistic formulations [14]; some optimize using conventional alternating minimization, others use convex-concave procedures [11].

Gehler and Chapelle [15] have recently performed an interesting analysis of the MIL costs, where deterministic annealing (DA) was used to compute better local optima for several formulations. In the case of a previous mi-SVM formulation [2], annealing methods did not improve the performance significantly. A newly proposed algorithm, ALP-SVM, was also introduced, which used a preset parameter defining the fixed ratio of witnesses – the true positive instances in a positive bag. Excellent results were obtained with this *witness rate* parameter set to the correct value. However, in practice it is unclear whether this can be known beforehand and whether it is stationary across different bags. In principle, the witness rate should also be estimated, and this learning stage partially account for the non-convexity of the MIL problem. It remains however unclear whether the observed performance variations are caused by non-convexity or by other modeling aspects.

Although performance considerations have hindered the application of MIL to practical problems, the methodology has started to gain momentum recently [4, 16]. The success of the Latent SVM for person detection [4] shows that a standard MIL procedure (the reformulation of the alternating minimization MI-SVM algorithm in [2]) can achieve good results if properly initialized. However, proper initialization of MIL remains elusive in general, as it often requires engineering experience with the individual problem structure. Therefore, it is still of broad interest to develop an initialization-independent formulation for MIL. Recently Li et al. [17] proposed a convex instance-level MIL algorithm based on multiple kernel learning, where one kernel was used for each possible combination of instances. This creates an exponential number of constraints and requires a cutting-plane solver. Although the formulation is convex, its scalability drops significantly for bags with many instances.

In this paper we make an alternative attempt towards a convex formulation: we establish that non-convex MIL constraints can be recast reliably into convex constraints on the likelihood ratio between the positive and negative classes for each instance. We transform the multiple-instance learning problem into a convex joint estimation of the likelihood ratio function and the likelihood ratio values on training instances. The choice of the jointly convex loss function is rich, remarkably at least from a family of f-divergences. Theoretically, we prove consistency results for likelihood ratio estimation, thus showing that f-divergence loss functions upper bound the classification 0-1 loss tightly, unless the likelihood is very large.

A support vector regression scheme is implemented to estimate the likelihood ratio, and it is shown to separate positive and negative instances well. However, determining the correct threshold for instance classification from the training set remain non-trivial. To address this problem, we propose a post-processing step based on a bag classifier computed as a linear combination of likelihood ratios. While this is shown to be suboptimal in synthetic experiments, it still achieves state-of-the-art results in practical datasets, demonstrating the vast potential of the proposed approach.

## 2 Convex Reformulation of the Multiple Instance Constraint

Let us consider a learning problem with $n$ training instances in total, $n_+$ positive and $n_-$ negative. In negative bags, every instance is negative, hence we do not separately define such bags – instead we directly work with the instances. Let $\mathcal{B} = \{B_1, B_2, \ldots, B_k\}$ be positive bags and $\mathcal{X}^+ = \{x_1^+, x_2^+, \ldots, x_{n_+}^+\}$, $\mathcal{X}^- = \{x_1^-, x_2^-, \ldots, x_{n_-}^-\}$ be the training input, where each $x_i$ belongs to a positive bag $B_j$ and each $x_i^-$ is a negative instance. The goal of multiple instance learning is, given $\{\mathcal{X}^+, \mathcal{X}^-, \mathcal{B}\}$, to learn a decision rule, $\text{sign}(f(x))$, to predict the label $\{+1, -1\}$ for the test instance $x$.

The MIL problem can be characterized by two properties. 1) **negative-exclusion**: if none of the instances in a bag is positive, the bag is not positive. 2) **positive-identifiability**: if one of the instances in the bag is positive, the bag is positive. These properties are equivalent to a constraint $\max_{x_i \in B_j} f(x_i) \geq 0$ on positive bags. This constraint is not convex since the negative max function is concave. Reformulation into a sum constraint such as $\sum f(x) \geq 0$ would be convex, when

$f(x) = w^T x$ is linear [6]. However, this hardly retains **positive-identifiability**, since if there is only one $x_i$ with $f(x_i) > 0$, this can be superseded by other instances with $f(x_i) < 0$. Apparently, the distinction between the sum and the max operations is significant and difficult to ignore in this context.

However, in this paper we show that if MIL conditions are formulated as constraints on the likelihood ratio, convexity can be achieved. For example, the constraint:

$$\sum_{x_i \in B_j} \frac{\Pr(y = 1|x_i)}{\Pr(y = -1|x_i)} > |B_j| \tag{1}$$

can ensure *both* of the MIL properties. **Positive-identifiability** is satisfied when $\Pr(y = 1|x_i) \geq \frac{|B_i|}{|B_i|+1}$ or equivalently, when the positive examples all have very large margin.

When the size of the bag is large, the assumption $\Pr(y = 1|x_i) > \frac{|B_j|}{|B_j|+1}$ can be too strong. Therefore, we exploit large deviation bounds to reduce the quantity $|B_j|$, such that $\Pr(y = 1|x_i)$ does not have to be very large to satisfy the constraint. Intuitively, if the examples are not very ambiguous, i.e. $\Pr(y = 1|x_i)$ is not close to $1/2$, then likelihood ratio sums on negative examples can become much smaller, hence we can adopt a significantly lower threshold at some degree of violation of the **negative-exclusion** property. To this end, a common assumption is the low label noise [18, 19]:

$$M_\beta : \exists c > 0, \forall \epsilon, \Pr(0 < |\Pr(y = 1|x_i) - \frac{1}{2}| \leq \epsilon) \leq c\epsilon^\beta.$$

This assumes that the posterior $\Pr(y = 1|x_i)$ is usually not very close to $1/2$, meaning that most examples are not very ambiguous, which is usually reasonable. In [18, 19, 20], a number of results have been obtained implying that classifiers learned under this assumption converge to the Bayes error much faster than the conventional empirical process rate $O(n^{-1/2})$ of most standard classifiers, and can be as fast as $O(n^{-1})$. These theoretical results show that low label noise assumptions indeed supports learning with fewer observations.

Assuming $M_\beta$ holds, we prove the following result which allows us to relax the hard constraint (1):

**Theorem 1** $\forall \delta > 0$, *for each $x_i$ in a bag $B_j$, assume $y_i$ is drawn i.i.d. from the distribution $\Pr_{B_j}(y_i|x_i)$ that satisfies $M_\beta$. If all instances $x_i \in B_j$ are negative, then the probability that*

$$\sum_{x_i \in B_j} \frac{\Pr(y = 1|x_i)}{\Pr(y = -1|x_i)} \geq \frac{\beta + 4}{2(\beta + 1)(\beta + 2)}|B_j| + \sqrt{\frac{4\beta + 1}{2(\beta + 1)^2(2\beta + 3)}|B_j|\log 1/\delta} + \frac{\log 1/\delta}{3} \tag{2}$$

*is at most $\delta$.*

The proof is given in an accompanying technical report [21]. From Theorem 1, we could weaken the constraint (1) to obtain constraint (2) and still ensure **negative-exclusion** with probability $1 - \delta$. When $\beta$ is large, the reduction is significant. For example, for $\beta = 2$ and $\delta = 0.05$, the right-hand side of (2) is approximately $\frac{1}{4}|B_i| + \sqrt{\frac{3}{14}|B_i|} + 1$, which is an important decrease over $|B_i|$, whenever $|B_i| \geq 3$. Note that the i.i.d. assumption in Theorem 1 applies to each bag. Different bags can have different label distributions. This is often a significantly weaker assumption than the ones based on global i.i.d. of labels [8].

## 3  Likelihood Ratio Estimation

To estimate the likelihood ratio, one possibility would be to use kernel methods as nonparametric estimators over a RKHS. This approach was taken in [22], where predictions of the ratio provided a variational estimate of an $f$-divergence (or Ali-Silvey divergence) between two distributions. The formulation is powerful, yet not immediately applicable here. In our case, because of the uncertainty in the positive examples, $\Pr(y = 1|x)$ is not observed but has to be estimated. Therefore we need to optimize jointly as $\min_{f, \Pr(y=1|x)} D(f, \Pr(y = 1|x)) + \lambda||f||^2$ with loss function $D(f, g)$. This optimization would not be convex if a framework in [22] were taken.

The requirement to estimate two sets of variables simultaneously (e.g. $f$ and $\Pr(y = 1|x)$ here), is one of the major difficulties in turning multiple-instance learning into a convex problem. Approaches based on classification-style loss functions lead to non-convex optimization [2, 13]. However, since we are outside a classification setting, we can optimize over divergence measures $D_\phi(f, g)$ that are convex w.r.t. both $f$ and $g$. These measures are common. For example, the $f$-divergence family that includes many statistical distances, satisfies the following properties [23]:

$$L_1 : D(x, y) = \sum_i |x_i - y_i|; \chi^2 : D(x, y) = \sum_i \frac{(x_i - y_i)^2}{x_i};$$
$$\text{Kullback-Leibler} : D(x, y) = \sum_i x_i \log x_i - x_i \log y_i - x_i + y_i; \qquad (3)$$
$$\text{Symmetric Kullback-Leibler} : D(x, y) = \sum_i (y_i - x_i) \log y_i + (x_i - y_i) \log x_i - x_i + y_i$$

In principle, any of the measures given above can be used to estimate the likelihood ratio.

An important issue is the relationship between the likelihood ratio estimation and our final goal: binary classification. In [20], the authors give necessary and sufficient conditions for Bayes consistent learners by minimizing the mean of a surrogate loss function of the data. In this paper we extend these results to loss functions for likelihood ratio estimation. Let $R(f) = P(\text{sign}(y) \neq \text{sign}(f(x) - 1))$ be the 0-1 risk of a likelihood estimator $f$, with classification rule given by $\text{sign}(f(x) \geq 1)$. The Bayes risk is then $R^* = \inf_f R(f)$.

For a generic loss function $C(\alpha, \eta)$, let $\eta = \Pr(y = 1|x)$, we can define the C-risk as $R_C(f) = \mathbb{E}(C(f, \eta))$ and $R_C^* = \inf_f R_C(f)$. Our goal is to bound the excess 0-1 risk $R(f) - R^*$ by the excess-C risk $R_C(f) - R_C^*$, so that minimizing the excess-C risk can be converted into minimizing the classification loss. Let us further define the optimal conditional risk as $H(\eta) = \inf_{\alpha \in \mathbb{R}} C(\alpha, \eta)$, and $H^-(\eta) = \inf_{\alpha, (\alpha-1)(2\eta-1) \leq 0} C(\alpha, \eta)$. We say $C(\alpha, \eta)$ is *classification-calibrated* if for any $\eta \neq 1/2$, $H^-(\eta) > H(\eta)$. Then, we define the $\psi$-transform of $C(\alpha, \eta)$ as $\psi(\theta) = \tilde{\psi}^{**}(\theta)$, where $\tilde{\psi}(\theta) = H^-(\frac{1+\theta}{2}) - H(\frac{1+\theta}{2}), \theta \in [-1, 1]$, and $g^{**}$ is the Fenchel-Legendre biconjugate of $g$, which is essentially the largest convex lower bound of $g$ [20].

The difference between likelihood ratio estimation and the classification setting is in the asymmetric scaling of the loss function for positive and negative examples. Let $\psi_- = \psi(-x)$, $R_-(f) = \Pr(y = -1, f(x) > 1)$, $R_-^* = \inf_f R_-(f)$, $R_+(f) = \Pr(y = 1, f(x) < 1)$ and $R_+^* = \inf_f R_+(f)$ be the risk and Bayes risks on negative and positive examples, respectively. It is easy to prove that $R(f) - R^* = R_-(f) - R_-^* + R_+(f) - R_+^*$. We derived the following theorem:

**Theorem 2** *a) For any nonnegative loss function $C(\alpha, \eta)$, any measurable $f : \mathcal{X} \to \mathbb{R}$, and any probability distribution on $\mathcal{X} \times \{\pm 1\}$, $\psi_-(R_-(f) - R_-^*) + \psi(R_+(f) - R_+^*) \leq R_C(f) - R_C^*$. b) The following conditions are equivalent: (1) C is classification-calibrated; (2) For any sequence $(\theta_i)$ in $[0, 1]$, $\psi(\theta_i) \to 0$ if and only if $\theta_i \to 0$; (3) For every sequence of measurable functions $f_i : \mathcal{X} \to \mathbb{R}$ and every probability distribution on $\mathcal{X} \times \{\pm 1\}$, $R_C(f_i) \to R_C^*$ implies $R(f_i) \to R^*$.*

The proof is given in an accompanying technical report [21]. This suggests that if $\psi$ is well-behaved, minimizing $R_C(f)$ still gives a reasonable surrogate for the classification risk. Compared against Theorem 3 in [20] which has the form $\psi(R(f) - R^*) \leq R_C(f) - R_C^*$, the difference here stems from the different loss transforms used for the positive and the negative examples.

We consider an $f$-divergence of the likelihood as the loss function, i.e., $C(\alpha, \eta) = D(\alpha, \frac{\eta}{1-\eta})$, where $\frac{\eta}{1-\eta}$ is the likelihood ratio when the $\Pr(y = 1|x) = \eta$. From convexity arguments, it can be easily seen that $H(\eta) = C(\frac{\eta}{1-\eta}, \eta) = 0$ and $H^-(\eta) = D(1, \frac{\eta}{1-\eta})$, therefore $\tilde{\psi}(\theta) = D(1, \frac{1+\theta}{1-\theta})$. The $\psi$ for all the loss functions listed in (4) can be computed accordingly. In fig. 3 (a) we show the $\psi(\theta)$ of $L_1$ and the KL-divergence from (4) and compare it against the hinge loss (where $\tilde{\psi}(\theta) = |\theta|$ [20]) used for SVM classification. It could be seen that our approximation of the classification loss is accurate when $\Pr(y_i = 1|x_i)$ is small. However, likelihood estimation would severely penalize the misclassified positive examples with large $\Pr(y_i = 1|x_i)$. This suggests that for the joint estimation of $f$ and $\Pr(y_i = 1|x_i)$, the optimizer would tend to make $\Pr(y_i = 1|x_i)$ smaller, in order to avoid high penalties, as shown in fig. 1(b).

In fig. 1(a) we plot $\psi$ functions for different losses. We prefer an $L_1$ measure as it is closer to the classification hinge loss, at least for the negative examples. In the end we solve the nonparametric function estimation in RKHS using an epsilon-insensitive $L_1$ loss, which can be reformulated as

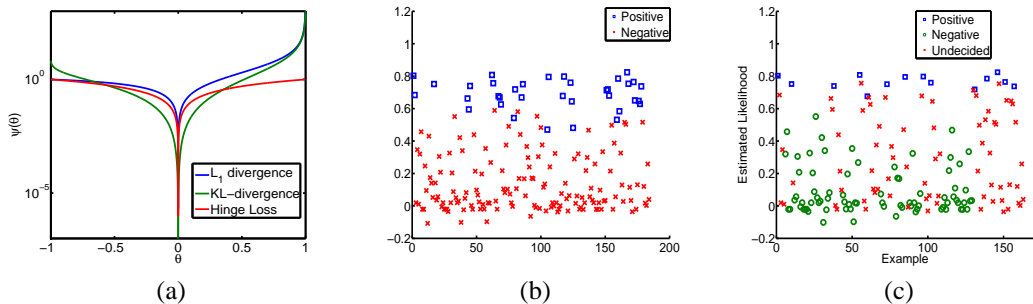

Figure 1: Loss functions and their influence on the estimation bias. **(a)** The function $\psi$ appearing in the losses used for likelihood estimation ($L_1$, KL-divergence) is similar to the hinge loss when $\theta > 0$; however it goes to infinity as $\theta$ approaches $1$. This deviation essentially means the surrogate loss is going to be extremely large if an example with very large $\Pr(y_i = 1 | x_i)$ is misclassified. **(b)** Example estimated likelihood for a synthetic example. The estimated likelihood is biased towards smaller values. However, with a fully labeled training set, the threshold can still be obtained. **(c)** If we only know the label of the negative examples (blue) and the maximal positive example (red), determining the optimal threshold becomes non-trivial.

support vector regression on the conditional likelihood, with the additional MIL constraints in (2):

$$\min_{f, \eta^+} \quad \sum_{x_j^+} \max(|f(x_j^+) - \eta_j^+| - \epsilon, 0) + \sum_{x_j^-} \max(|f(x_j^-)| - \epsilon, 0) + \lambda ||f||^2$$

$$\text{s.t.} \qquad\qquad\qquad \sum_{x_j^+ \in B_i} \eta_j^+ \geq D_i, \eta_j^+ \geq 0 \qquad\qquad\qquad\qquad (4)$$

where $||f||^2$ is the RKHS norm; $D_i$ is a constant for each bag and can be determined from Theorem 1, with appropriately chosen values for constants $\beta$ and $\delta$; $\eta_i^+$ is an estimate of $\frac{Pr(y=1|x_i^+)}{Pr(y=-1|x_i^+)}$ for the training set. In this paper we use $\beta = 2$ and $\delta = 0.05$, which gives the estimate of the bound for each bag as $D_i = \frac{1}{4}|B_i| + \sqrt{\frac{3}{14}|B_i|} + 1$, when $B_i \geq 3$ and $D_i = |B_i|$ when $|B_i| < 3$.

It can be proved that optimization problem (4) is jointly convex in both arguments. A standard representer theorem [24] would convert it to an optimization on vectors, which we omit here. The problem can be solved by different methods. The one easiest to implement is the alternating minimization between solving for the SVM and projecting on the constraint sets given by $\sum_{x_j^+ \in B_i} y_j^+ \geq D_i$ and $y_j^+ \geq 0$. As this can turn out to be slow for large datasets, approaches such as the dual SMO or primal subgradient projection algorithms (in the case of linear SVM) can be used. In this paper we implement the alternating minimization approach, which is provably convergent since the optimization problem (4) is convex. In the accompanying technical report [21] we derive an SMO algorithm based on the dual of (4) and characterize the basic properties of the optimization problem.

## 4 Bag and Instance Classification

If the likelihood ratio is obtained using an unbiased estimator, a decision rule based on $\text{sign}(f(x) \geq 1)$ should give the optimal classifier. However as previously argued, the joint estimation on $f$ and $\eta^+$ introduces a bias which is not always easy to identify. In positive bags, it is unclear whether an instance should be labeled positive or negative, as long as it does not contribute significantly to the classification error of its bag (fig. 3(b),(c)). In the synthetic experiments, we noticed that knowledge of the correct threshold would make the algorithm outperform competitors by a large margin (fig. 2). This means that based on the learned likelihood ratio, the positive examples are usually well separated from the negative ones. Developing a theory that would advance these aspects remains a promising avenue for future work. The main difficulty stems from the compound source of bias which arises from both the estimation of $\eta^+$ and the loss minimization over $\eta^+$ and $f$.

Here we propose a partial solution. Instead of directly estimating the threshold, we learn a linear combination of instance likelihood ratios to classify the bag. First, we sort the instance likelihood ratios for each bag into a vector of length $\max_i |B_i|$. We append 0 to bags that do not have enough

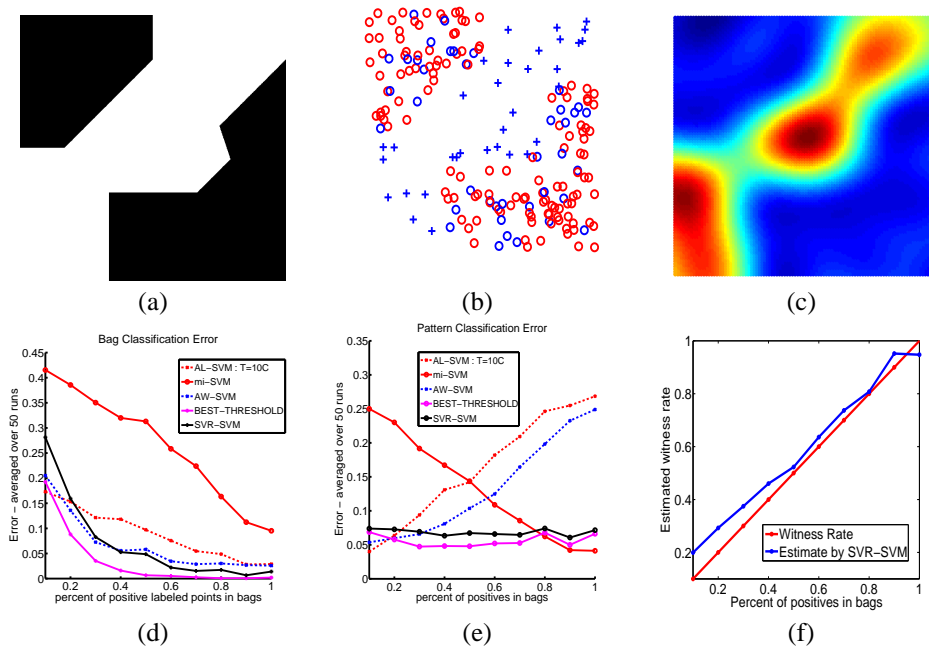

Figure 2: Synthetic dataset (best viewed in color). **(a)** The true decision boundary. **(b)** Training points at $40\%$ witness rate. **(c)** The learned regression function. **(d)** Bag misclassification rate of different algorithms. **(e)** Instance misclassification rate of different algorithms. **(f)** Estimated witness rate and true witness rate.

instances. Under this representation, bag classification turns into a standard binary decision problem where a vector and a binary label is given for each bag, and a linear SVM is learned to solve the problem. If we were to classify only the likelihood ratio on the first instance, this procedure would reduce to simple thresholding. We instead leverage information in the entire bag, aiming to constrain the classifier to learn the correct threshold. In this linear SVM setting, regularization never helps in practice and we always fix $C$ to very large values. Effectively no parameter tuning is needed.[1]

To classify instances, a threshold is still necessary. In the current system, we follow a simple approach and take the mean between two instances: the one with the highest likelihood among training bags that are predicted negative by the bag classifier, and the lowest scored one among instances in positive bags with a score higher than the previous one. This approach is derived from the basic MIL assumption that all instances in a negative bag are negative.

Based on instance classification we could also estimate the witness rate of the dataset. This is computed as the ratio of positively classified instances and the total number of instances in the positive bags of the training set. Since our algorithm automatically adjusts to different witness rates, this estimate offers quantitative insight as to whether MIL should be used. For instance, if the witness rate is $100\%$, it may be more effective to use a conventional learning approach.

# 5 Experiments

## 5.1 Synthetic Data

We start with an experiment on the synthetic dataset of [15], where the controlled setting helps understanding the behavior of the proposed algorithm. This is a 2-D dataset with the actual decision boundary shown in fig. 2 (a). The positive bags have a fraction of points sampled uniformly from the white region and the rest sampled uniformly from the black region. An example of the sample at $40\%$ witness rate is shown in fig. 2 (b). In this figure, the plotted instance labels are the ones of their bags – indeed, one could notice many positive (blue) instances in the negative (red) region.

Table 1: Performance of various MIL algorithms on weak labeling benchmarks. The best result on each dataset is shown in bold. The second group of algorithms either not provide instance labels (MI-Kernel and miGraph) or require a parameter that can be difficult to tune (ALP-SVM). SVR-SVM appears to give consistent results among algorithms that provide instance labels. The row denoted "Est. WR" gives the estimated witness rates of our method.

| Algorithm | Musk-1 | Musk-2 | Elephant | Tiger | Fox |
|---|---|---|---|---|---|
| CH-FD | 88.8 | 85.7 | 82.4 | 82.2 | 60.4 |
| EMDD | 84.9 | 84.8 | 78.3 | 72.1 | 56.1 |
| mi-SVM | 87.4 | 83.6 | 82.2 | 78.4 | 58.2 |
| MI-SVM | 77.9 | 84.3 | 81.4 | 84.0 | 57.8 |
| MICA | 84.4 | **90.5** | 82.5 | 82.0 | 62.0 |
| AW-SVM | 85.7 | 83.8 | 82.0 | 83.0 | 63.5 |
| Ins-KI-SVM | 84.0 | 84.4 | 83.5 | 82.9 | 63.4 |
| MI-Kernel | $88.0 \pm 3.1$ | $89.3 \pm 1.5$ | $84.3 \pm 1.6$ | $84.2 \pm 1.9$ | $60.3 \pm 1.0$ |
| miGraph | $\mathbf{88.9} \pm 3.3$ | $90.3 \pm 2.6$ | $\mathbf{86.8} \pm 0.7$ | $\mathbf{86.0} \pm 2.8$ | $61.6 \pm 1.6$ |
| ALP-SVM | 86.3 | 86.2 | 83.5 | 86.0 | **66.0** |
| SVR-SVM | $87.9 \pm 1.7$ | $85.4 \pm 1.8$ | $\mathbf{85.3} \pm 2.8$ | $79.8 \pm 3.4$ | $\mathbf{63.0} \pm 3.5$ |
| Est. WR | 100 % | 89.5 % | 37.8 % | 42.7 % | 100 % |

In order to test the effect of witness rates, 10 different types of datasets are created by varying the rates over the range $0.1, 0.2, \ldots, 1$. In this experiment we fix the hyperparameters $C = 5$ and use a Gaussian kernel with $\sigma = 1$. We show a trained likelihood ratio function in fig. 2 (c), estimated on the dataset shown in fig. 2 (b). Under the likelihood ratio, the positive examples are well separated from negatives. This illustrates how our proposed approach converts multiple-instance learning into the problem of deciding a one-dimensional threshold.

Complete results on datasets with different witness rates are shown in fig. 2 (d) and (e). We give both bag classification and instance classification results. Our approach is referred to as SVR-SVM. BEST THRESHOLD refers to a method where the best threshold was chosen based on the full knowledge of training/test instance labels, i.e., the optimal performance our likelihood ratio estimator can achieve. Comparison is done with two other approaches, the mi-SVM in [2] and the AW-SVM from [15]. SVR-SVM generally works well when the witness rate is not very low. From instance classification, one can see that the original mi-SVM is only competitive when the witness rate is near 1 – this situation is close to a supervised SVM. With a deterministic annealing approach in [15], AW-SVM and mi-SVM perform quite the opposite – competitive when the witness rate is small but degrade when this is large. Presumably this is because deterministic annealing is initialized with the apriori assumption that datasets are multiple-instance i.e. has a small witness rate [15]. When the witness rate is large, annealing does not improve performance. On the contrary, the proposed SVR-SVM does not appear to be affected by the witness rate. With the same parameters used across all the experiments, the method self-adjusts to different witness rates. One could see the effect especially in fig. 2 (e): regardless of the witness rate, the instance error rate remains roughly the same. However, this is still inferior to our model based on the best threshold, which indicates that important room for improvement exists.

## 5.2  MIL Datasets

The algorithm is evaluated on a number of popular MIL benchmarks. We use the common experimental setting, based on 10-fold cross-validation for parameter selection and we report the test results averaged over 10 trials. The results are shown in Table 1, together with other competitive methods in from the literature [12, 15, 10] (for some of these methods standard deviation estimates are not available).

In our tests, the proposed SVR-SVM gives consistently good results among algorithms that provide instance-level labels. The only atypical case is *Tiger*, where the algorithm underperforms other methods. Overall, the performance of SVR-SVM is slightly worse than miGraph and ALP-SVM. But we note that results in ALP-SVM are obtained by tuning the witness rate to the optimal value, which may be difficult in practical settings. The slightly lower performance compared to miGraph suggests that we may be inferior in the bag classification step, which we already know is suboptimal.

Table 2: Results from *20 Newsgroups*. The best result on each dataset is shown in bold, pairwise *t*-tests are performed to determine if the differences are statistically significantly. miGraph is dominating in 10 datasets, whereas SVR-SVM is dominating in 14.

| Dataset | MI-Kernel | miGraph [10] | miGraph (web) | SVR-SVM | Est. WR |
|---|---|---|---|---|---|
| alt.atheism | $60.2 \pm 3.9$ | $65.5 \pm 4.0$ | $82.0 \pm 0.8$ | $\mathbf{83.5} \pm 1.7$ | 1.83 % |
| comp.graphics | $47.0 \pm 3.3$ | $77.8 \pm 1.6$ | $\mathbf{84.3} \pm 0.4$ | $\mathbf{85.2} \pm 1.5$ | 5.19 % |
| comp.windows.misc | $51.0 \pm 5.2$ | $63.1 \pm 1.5$ | $\mathbf{70.1} \pm 0.3$ | $66.9 \pm 2.6$ | 2.23 % |
| comp.ibm.pc.hardware | $46.9 \pm 3.6$ | $59.5 \pm 2.7$ | $\mathbf{79.4} \pm 0.8$ | $70.3 \pm 2.8$ | 2.42 % |
| comp.sys.mac.hardware | $44.5 \pm 3.2$ | $61.7 \pm 4.8$ | $\mathbf{81.0} \pm 0$ | $78.0 \pm 1.7$ | 4.58 % |
| comp.window.x | $50.8 \pm 4.3$ | $69.8 \pm 2.1$ | $79.4 \pm 0.5$ | $\mathbf{83.7} \pm 2.0$ | 5.36 % |
| misc.forsale | $51.8 \pm 2.5$ | $55.2 \pm 2.7$ | $71.0 \pm 0$ | $\mathbf{72.3} \pm 1.2$ | 4.29 % |
| rec.autos | $52.9 \pm 3.3$ | $72.0 \pm 3.7$ | $\mathbf{83.2} \pm 0.6$ | $78.1 \pm 1.9$ | 2.75 % |
| rec.motorcycles | $50.6 \pm 3.5$ | $64.0 \pm 2.8$ | $70.9 \pm 2.7$ | $\mathbf{75.6} \pm 0.9$ | 2.86 % |
| rec.sport.baseball | $51.7 \pm 2.8$ | $64.7 \pm 3.1$ | $75.0 \pm 0.6$ | $\mathbf{76.7} \pm 1.4$ | 4.31 % |
| rec.sport.hockey | $51.3 \pm 3.4$ | $85.0 \pm 2.5$ | $\mathbf{92.0} \pm 0$ | $89.3 \pm 1.6$ | 6.52 % |
| sci.crypt | $56.3 \pm 3.6$ | $69.6 \pm 2.1$ | $\mathbf{70.1} \pm 0.8$ | $\mathbf{69.7} \pm 2.5$ | 3.22 % |
| sci.electronics | $50.6 \pm 2.0$ | $87.1 \pm 1.7$ | $\mathbf{94.0} \pm 0$ | $91.5 \pm 1.0$ | 4.29 % |
| sci.med | $50.6 \pm 1.9$ | $62.1 \pm 3.9$ | $72.1 \pm 1.3$ | $\mathbf{74.9} \pm 1.9$ | 5.23 % |
| sci.space | $54.7 \pm 2.5$ | $75.7 \pm 3.4$ | $79.4 \pm 0.8$ | $\mathbf{83.2} \pm 2.0$ | 3.64 % |
| soc.religion.christian | $49.2 \pm 3.4$ | $59.0 \pm 4.7$ | $75.4 \pm 1.2$ | $\mathbf{83.2} \pm 2.7$ | 3.30 % |
| talk.politics.guns | $47.7 \pm 3.8$ | $58.5 \pm 6.0$ | $\mathbf{72.3} \pm 1.0$ | $\mathbf{73.7} \pm 2.6$ | 3.23 % |
| talk.politics.mideast | $55.9 \pm 2.8$ | $73.6 \pm 2.6$ | $75.5 \pm 1.0$ | $\mathbf{80.5} \pm 3.2$ | 3.88 % |
| talk.politics.misc | $51.5 \pm 3.7$ | $70.4 \pm 3.6$ | $\mathbf{72.9} \pm 2.4$ | $\mathbf{72.6} \pm 1.4$ | 2.82 % |
| talk.religion.misc | $55.4 \pm 4.3$ | $63.3 \pm 3.5$ | $67.5 \pm 1.0$ | $\mathbf{71.9} \pm 1.9$ | 2.87 % |

## 5.3 Text Categorization

The text datasets are taken from [10]. These data have the benefit of being designated to have a small witness rate. Thus they serve as a better MIL benchmark compared to the previous ones. These are derived from the *20 Newsgroups* corpus, with 50 positive and 50 negative bags for each of the 20 news categories. Each positive bag has around $3\%$ witness rate. We run 10-fold cross validation 10 times on each dataset and compute the average accuracy and standard deviations, $C$ is fixed to 100, $\epsilon$ to 0.2. Authors of [10] reported recent results for this dataset on their website, which are vastly superior than the ones reported in the paper. Therefore, in Table 2 we included both results in the comparison, identified as miGraph (paper) and miGraph (website), respectively.

Our SVR-SVM performs significantly better than MI-Kernel and miGraph (paper). It is comparable with miGraph (web), and offers a marginal improvement. It is interesting that even though we use a suboptimal second step, SVR-SVM fares well with the state-of-the-art. This shows the potential of methods based on likelihood ratio estimators for multiple instance learning.

## 6 Conclusion

We have proposed an approach to multiple-instance learning based on estimating the likelihood ratio between the positive and the negative classes on instances. The MIL constraint is reformulated into a convex constraint on the likelihood ratio where a joint estimation of *both* the function *and* the target ratios on the training set is performed. Theoretically we justify that learning the likelihood ratio is Bayes-consistent and has desirable excess loss transform properties. Although we are not able to find the optimal classification threshold on the estimated ratio function, our proposed bag classifier based on such ratios obtains state-of-the-art results in a number of difficult datasets. In future work, we plan to explore transductive learning techniques in order to leverage the information in the learned ratio function and identify better threshold estimation procedures.

**Acknowledgements**

This work is supported, in part, by the European Commission, under a Marie Curie Excellence Grant MCEXT-025481.

## Footnotes

[1]We have also experimented with a uniform threshold based on probabilistic estimates, as well as with predicting an instance-level threshold. While the former tends to underfit, the latter overfits. Our bag-level classifier targets an intermediate level of granularity and turns out to be the most robust in our experiments.

# References

[1] Dietterich, T.G., Lathrop, R.H., Lozano-Perez, T.: Solving the multiple-instance problem with axis-parallel rectangles. Artificial Intelligence **89** (1997) 31–71

[2] Andrews, S., Tsochantaridis, I., Hofmann, T.: Support vector machines for multiple-instance learning. In: NIPS. (2003) 561–568

[3] Maron, O., Lozano-Pérez, T.: A framework for multiple-instance learning. In: NIPS. (1998) 570–576

[4] Felzenszwalb, P.F., McAllester, D.A., Ramanan, D.: A discriminatively trained, multiscale, deformable part model. In: CVPR. (2008)

[5] Russell, B.C., Torralba, A., Murphy, K.P., Freeman, W.T.: Labelme: A database and web-based tool for image annotation. IJCV **77**(1-3) (2008) 157–173

[6] Cour, T., Sapp, B., Nagle, A., Taskar, B.: Talking pictures: Temporal grouping and dialog-supervised person recognition. In: CVPR. (2010)

[7] Zeisl, B., Leistner, C., Saffari, A., Bischof, H.: On-line semi-supervised multiple-instance boosting. In: CVPR. (2010)

[8] Gärtner, T., Flach, P.A., Kowalczyk, A., Smola, A.J.: Multi-instance kernels. In: ICML. (2002)

[9] Tao, Q., Scott, S., Vinodchandran, N.V., Osugi, T.T.: Svm-based generalized multiple-instance learning via approximate box counting. In: ICML. (2004)

[10] Zhou, Z.H., Sun, Y.Y., Li, Y.F.: Multi-instance learning by treating instances as non-i.i.d. samples. In: ICML. (2009)

[11] Cheung, P.M., Kwok, J.T.: A regularization framework for multiple-instance learning. In: ICML. (2006) 193–200

[12] Fung, G., Dandar, M., Krishnapuram, B., Rao, R.B.: Multiple instance learning for computer aided diagnosis. In: NIPS. (2007)

[13] Mangasarian, O., Wild, E.: Multiple instance classification via successive linear programming. Journal of Optimization Theory and Applications **137** (2008) 555–568

[14] Zhang, Q., Goldman, S.A., Yu, W., Fritts, J.E.: Content-based image retrieval using multiple-instance learning. In: ICML. (2002) 682–689

[15] Gehler, P., Chapelle, O.: Deterministic annealing for multiple-instance learning. In: AISTATS. (2007)

[16] Dollár, P., Babenko, B., Belongie, S., Perona, P., Tu, Z.: Multiple component learning for object detection. In: ECCV. (2008)

[17] Li, Y.F., Kwok, J.T., Tsang, I.W., Zhou, Z.H.: A convex method for locating regions of interest with multi-instance learning. In: ECML. (2009)

[18] Mammen, E., Tsybakov, A.B.: Smooth discrimination analysis. Annals of Statistics **27** (1999) 1808–1829

[19] Tsybakov, A.B.: Optimal aggregation of classifiers in statistical learning. Annals of Statistics **32** (2004) 135–166

[20] Bartlett, P., Jordan, M.I., McAulliffe, J.: Convexity, classification and risk bounds. Journal of American Statistical Association **101** (2006) 138–156

[21] Li, F., Sminchisescu, C.: Convex multiple instance learning by estimating likelihood ratio. Technical report, Institute for Numerical Simulation, University of Bonn (November 2010)

[22] Nguyen, X., Wainwright, M., Jordan, M.I.: Estimating divergence functionals and the likelihood ratio by penalized convex risk minimization. In: NIPS. (2007)

[23] Liese, F., Vajda, I.: Convex Statistical Distances. Teubner VG (1987)

[24] Hofmann, T., Schölkopf, B., Smola, A.J.: Kernel methods in machine learning. The Annals of Statistics **36** (2008) 1171–1220

